# Fast detection of multiple change-points shared by many signals using group LARS

**Jean-Philippe Vert and Kevin Bleakley**
Mines ParisTech CBIO, Institut Curie, INSERM U900
{firstname.lastname}@mines-paristech.fr

## Abstract

We present a fast algorithm for the detection of multiple change-points when each is frequently shared by members of a set of co-occurring one-dimensional signals. We give conditions on consistency of the method when the number of signals increases, and provide empirical evidence to support the consistency results.

## 1 Introduction

Finding the place (or time) where most or all of a set of one-dimensional signals (or *profiles*) jointly change in some specific way is an important question in several fields. A first common situation is when we want to find change-points in a multidimensional signal, for instance, we may want to automatically detect changes from *human speech* to *other sound* in a movie, based on data representation of features coming from both the audio and visual tracks [1]. Another important situation is when we are confronted with several 1-dimensional signals which we believe share common change-points, e.g., genomic profiles of several patients. The latter application is increasingly important in biology and medicine, in particular for the detection of copy-number variation along the genome, though it is also useful for microarray and genetic linkage studies [2]. The common thread in all of these is the search for data patterns shared by a set of patients at precise places on the genome; in particular, sudden changes in measurement. As opposed to the segmentation of multi-dimensional signals such as speech, the length of the signal (i.e., the number of probes along the genome) is fixed for a given technology while the number of signals (i.e., the number of patients) can increase. It is therefore of interest to develop method to identify multiple change-points shared by several signals which can benefit from increasing the number of profiles.

There exists a vast literature on the change-point detection problem [3, 4]. Here we focus on the problem of approximating a multidimensional signal by a piecewise-constant one, using quadratic error criteria. It is well-known that the optimal segmentation of a $p$-dimensional signal of length $n$ into $k$ segments can be obtained in $O(n^2 pk)$ by dynamic programming [5, 6, 7]. The quadratic complexity in $n^2$ is however prohibitive in applications such as genomics, where $n$ can be in the order of $10^5$ to $10^7$ with current technology. An alternative to such *global* procedures, which estimate change-points as solutions of a global optimization problem, are fast *local* procedures such as binary segmentation [8], which detect breakpoints by iteratively applying a method for single change-point detection to the segments obtained after the previous change-point is detected. While such recursive methods can be extremely fast, in the order of $O(np \log(k))$ when the single change-point detector is $O(np)$, quality of segmentation is questionable when compared with global procedures [9].

For $p = 1$ (a single signal), an interesting alternative to these global and local procedures is to express the optimal segmentation as the solution of a convex optimization problem, using the (convex) total variation instead of the (non-convex) number of jumps to penalize a piecewise-constant function, in order to approximate the original signal [10, 11]. The resulting piecewise-constant approximation of the signal, defined as the global minimum of the objective function, benefits from

theoretical guaranties in terms of correctly detecting change-points [12, 13], and can be implemented efficiently in $O(nk)$ or $O(n\log(n))$ [14, 12, 15].

In this paper we propose an extension of total-variation based methods for single signals to the multidimensional setting, in order to approximate a multidimensional signal by a piecewise constant signal with multiple change-points. We define the approximation as the solution of a convex optimization problem, which involves a quadratic approximation error penalized by the $\ell_1$ norm of increments of the function. The problem can be reformulated as a group LASSO problem, which we propose to solve approximately with a group LARS procedure [16]. Using the particular structure of the design matrix, we can find the first $k$ change-points in $O(npk)$, extending the method of [12] to the multidimensional setting.

Unlike most previous theoretical investigations of change-point methods, we are not interested in the case where the dimension $p$ is fixed and the length of the profiles $n$ increases, but in the opposite situation where $n$ is fixed and $p$ increases. Indeed, this corresponds to the case in genomics where, for example, $n$ would be the fixed number of probes used to measure a signal along the genome, and $p$ the number of samples or patients analyzed. We want to design a method that benefits from increasing $p$ in order to identify shared change-points, even though the signal-to-noise ratio may be very low within each signal. As a first step towards this question, we give conditions under which our method is able to consistently identify a single change-point as $p$ increases. We also show by simulation that our method is able to consistently identify multiple change-points, as $p \to +\infty$, validating its relevance in practical settings. To conclude, we present possible applications of the method in the study of copy number variations in cancer.

## 2   Notation

For any two integers $u \leq v$, let $[u, v]$ denote the interval $\{u, u+1, \ldots, v\}$. For any $u \times v$ matrix $M$ we note $M_{i,j}$ its $(i, j)$-th entry. $\|M\| = \sqrt{\sum_{i=1}^{u} \sum_{j=1}^{v} M_{i,j}^2}$ is its Frobenius norm (or Euclidean norm in the case of vectors). For any subsets of indices $A = \{a_1, \ldots, a_{|A|}\} \in [1, u]^{|A|}$ and $B = (b_1, \ldots, b_{|B|}) \in [1, v]^{|B|}$, we denote by $M_{A,B}$ the $|A| \times |B|$ matrix with entries $M_{a_i, b_j}$ for $(i, j) \in [1, |A|] \times [1, |B|]$. For simplicity we will use $\bullet$ instead of $[1, u]$ or $[1, v]$, i.e., $A_{i,\bullet}$ is the $i$-th row of $A$ and $A_{\bullet, j}$ is the $j$-th column of $A$. We note $\mathbf{1}_{u,v}$ the $u \times v$ matrix of ones, and $\mathbf{I}_p$ the $p \times p$ identity matrix.

## 3   Formulation

We consider $p$ profiles of length $n$, stored in an $n \times p$ matrix $Y$. The $i$-th profile $Y_{\bullet, i} = (Y_{1,i}, \ldots, Y_{n,i})$ is the $i$-th column of $Y$. We assume that each profile is a piecewise-constant signal corrupted by noise, and that change-points locations tend to be shared across profiles. Our goal is to detect these shared change-points, and benefit from the possibly large number $p$ of profiles to increase the statistical power of change-point detection.

When $p = 1$ (single profile), a popular method to find change-points in a signal is to approximate it by a piecewise constant function using total variation (TV) denoising [10], i.e., to solve

$$\min_{U \in \mathbb{R}^n} \| Y - U \|^2 + \lambda \sum_{i=1}^{n-1} | U_{i+1} - U_i | \,. \tag{1}$$

For a given $\lambda > 0$, the solution $U \in \mathbb{R}^n$ of (1) is piecewise-constant and its change-points are predicted to be those of $Y$. Adding penalties proportional to the $\ell_1$ ot $\ell_2$ norm of $U$ to (1) does not change the position of the change-points detected [11, 17], and the capacity of TV denoising to correctly identify change-points when $n$ increases has been investigated in [12, 13].

Here we propose to generalize TV denoising to multiple profiles by considering the following convex optimization problem, for $Y \in \mathbb{R}^{n \times p}$:

$$\min_{U \in \mathbb{R}^{n \times p}} \| Y - U \|^2 + \lambda \sum_{i=1}^{n-1} \| U_{i+1,\bullet} - U_{i,\bullet} \| \,. \tag{2}$$

The second term in (2) penalizes the sum of Euclidean norms of the increments of $U$, seen as a time-dependent multidimensional vector. Intuitively, this penalty will enforce many increments $U_{i+1,\bullet} - U_{i,\bullet}$ to collapse to 0, just like the total variation in (1). As a result the solution of (2) provides an approximation of the profiles $Y$ by a $n \times p$ matrix of piecewise-constant profiles $U$ which share change-points. In the following, we propose a fast algorithm to approximately solve (2) (Section 4), discuss theoretically whether the solution identifies correctly the change-points (Section 5), and provide an empirical evaluation of the method (Section 6).

## 4  Implementation

Although (2) is a convex optimization problem that can in principle be solved by general-purpose solvers [18], we are often working in dimensions that can reach millions, making this approach impractical. Moreover, we would ideally like to obtain solutions for various values of $\lambda$, corresponding to various numbers of change-points, in order to be able to select the optimal number of change-points using various statistical criteria. In the single profile case ($p = 1$), [14] proposed a fast coordinate descent-like method, [12] showed how to find the first $k$ change-points iteratively in $O(nk)$, and [15] proposed an $O(n \ln(n))$ method to find all change-points. However, none of these methods is applicable directly to the $p > 1$ setting since they all rely on specific properties of the $p = 1$ case, such as the fact that the solution is piecewise-affine in $\lambda$ and that the set of change-points is monotically decreasing with $\lambda$.

In order to propose a fast method to solve (2) in the $p > 1$ setting, let us first reformulate it as a group LASSO regression problem [16]. To this end, we make the change of variables $(\beta, \gamma) \in \mathbb{R}^{(n-1) \times p} \times \mathbb{R}^{1 \times p}$ given by:

$$\gamma = U_{1,\bullet} \,,$$
$$\beta_{i,\bullet} = U_{i+1,\bullet} - U_{i,\bullet} \quad \text{for } i = 1, \ldots, n-1 \,.$$

In other words $\beta_{i,j}$ is the jump between the $i$-th and the $(i + 1)$-th positions of the $j$-th profile. We immediately get an expression of $U$ as a function of $\beta$ and $\gamma$:

$$U_{1,\bullet} = \gamma \,,$$
$$U_{i,\bullet} = \gamma + \sum_{j=1}^{i-1} \beta_{j,\bullet} \quad \text{for } i = 2, \ldots, n \,.$$

This can be rewritten in matrix form as

$$U = \mathbf{1}_{n,1} \gamma + X\beta \,,$$

where $X$ is the $n \times (n - 1)$ matrix with entries $X_{i,j} = 1$ for $i > j$. Making this change of variable, we can re-express (2) as follows:

$$\min_{\beta \in \mathbb{R}^{(n-1) \times p}, \gamma \in \mathbb{R}^{1 \times p}} \| Y - X\beta - \mathbf{1}_{n,1}\gamma \|^2 + \lambda \sum_{i=1}^{n-1} \| \beta_{i,\bullet} \| \,. \tag{3}$$

For any $\beta \in \mathbb{R}^{(n-1) \times p}$, the minimum in $\gamma$ is reached for $\gamma = \mathbf{1}_{1,n}(Y - X\beta)/n$. Plugging this into (3), we get that the matrix of jumps $\beta$ is solution of

$$\min_{\beta \in \mathbb{R}^{(n-1) \times p}} \| \bar{Y} - \bar{X}\beta \|^2 + \lambda \sum_{i=1}^{n-1} \| \beta_{i,\bullet} \| \,, \tag{4}$$

where $\bar{Y}$ and $\bar{X}$ are obtained from $Y$ and $X$ by centering each column.

Equation 4 is a group LASSO problem, with a particular design matrix and particular groups of features. Since existing methods to exactly solve group LASSO regression problems remain difficult to apply here – in particular we do not want to store in memory the $n \times (n - 1)$ design matrix when $n$ is in the millions – we propose to approximate instead the solution of (4) with the *group LARS* strategy, which was proposed by [16] as a good approximation to the regularization path of the group LASSO. More precisely, the group LARS approximates the solution path of (4) with a piecewise-affine set of solutions, and iteratively finds change-points. While the original group LARS method requires storing and manipulation of the design matrix [16], which we can not afford here, we can extend technical results of [12] to show that the particular structure of the design matrix $\bar{X}$ allows efficient computation of matrix inverses and products.

**Lemma 1.** *For any $R \in \mathbb{R}^{n \times p}$, we can compute $C = \bar{X}^\top R$ in $O(np)$ time and memory.*

**Lemma 2.** *For any $A = (a_1, \ldots, a_{|A|})$, set of distinct indices with $1 \leq a_1 < \ldots < a_{|A|} \leq n$, the matrix $(\bar{X}_{\bullet,A}^\top \bar{X}_{\bullet,A})$ is invertible, and for any $|A| \times p$ matrix $R$, the matrix*

$$C = (\bar{X}_{\bullet,A}^\top \bar{X}_{\bullet,A})^{-1} R$$

*can be computed in $O(|A|p)$ time and memory.*

Proof of these results can be found in Supplementary Materials.

Algorithm 1 describes the fast group LARS method to approximately solve (4). At each subsequent iteration to find the next change-point, we follow steps 3–8 which have maximum complexity $O(np)$, resulting in $O(npk)$ complexity in time and $O(np)$ in memory to find the first $k$ change-points with the fast group LARS algorithm.

---

**Algorithm 1** Fast group LARS algorithm

---

**Require:** centered data $\bar{Y}$, number of breakpoints $k$.
 1: Initialize $r = \bar{Y}$, $\mathcal{A} = \emptyset$.
 2: **for** $i = 1$ to $k$ **do**
 3:     Compute $\hat{c} = \bar{X}^\top r$ using Lemma 1.
 4:     If $i = 1$, find the first breakpoint : $\hat{a} = \mathrm{argmin}_{j \in [1,n]} \| \hat{c}_{j,\bullet} \|$, $\mathcal{A} = \{\hat{a}\}$.
 5:     Descent direction: compute $w = (\bar{X}_{\mathcal{A},\bullet}^\top \bar{X}_{\mathcal{A},\bullet})^{-1} \hat{c}_{\mathcal{A},\bullet}$ using Lemma 2, then $u_{\mathcal{A}} = \bar{X}_{\mathcal{A}} w$ with cumulative sums, then $a = \bar{X}^\top u_{\mathcal{A}}$ using Lemma 1.
 6:     Descent step: for each $u \in [1,n] \setminus \mathcal{A}$, find if it exists the smallest positive solution $\alpha_u$ of the second-order polynomial in $\alpha$:

$$\| \hat{c}_{u,\bullet} - \alpha a_{u,\bullet} \|^2 = \| \hat{c}_{v,\bullet} - \alpha a_{v,\bullet} \|^2 \,,$$

   where $v$ is any element of $\mathcal{A}$.
 7:     Find the next breakpoint: $\hat{u} = \mathrm{argmin}_{[1,p] \setminus \mathcal{A}} \alpha_u$.
 8:     Update $\mathcal{A} = \mathcal{A} \cup \{\hat{u}\}$ and $r = r - a_{\hat{u}} u_{\mathcal{A}}$.
 9: **end for**

---

## 5 Theoretical analysis

In this section, we study theoretically to what extent the estimator (2) recovers correct change-points. The vast majority of existing theoretical results for offline segmentation and change-point detection consider the setting where $p$ is fixed (usually $p = 1$), and $n$ increases. This typically corresponds to a setting where we can sample a continuous signal with increasing density, and wish to locate more precisely the underlying change-points as the density increases.

Here we propose a radically different analysis, motivated by applications in genomics. Here, the length of profiles $n$ is fixed for a given technology, but the number of profiles $p$ can increase when more biological samples or patients are analyzed. The property we would like to study is then, for a given change-point detection method, whether increasing $p$ for fixed $n$ allows us to locate more precisely the change-points. While this simply translates our intuition that increasing the number of profiles should increase the statistical power of change-point detection, and while this property was empirically observed in [2], we are not aware of previous theoretical results in this setting.

### 5.1 Consistent estimation of a single change-point

As a first step towards the analysis of this "fixed $n$ increasing $p$" setting, let us assume that the observed centered profiles $\bar{Y}$ are obtained by adding noise to a set of profiles with a *single* shared change-point between positions $u$ and $u + 1$, for some $u \in [1, n-1]$. In other words, we assume that

$$\bar{Y} = \bar{X}\beta^* + W \,,$$

where $\beta^*$ is an $(n-1) \times p$ matrix of zeros except for the $u$-th row $\beta_{u,\bullet}^*$, and $W$ is a noise matrix whose entries are assumed to be independent and identically distributed with respect to a centered Gaussian

distribution with variance $\sigma^2$. In this section we study the probability that the first breakpoint found by our procedure is the correct one, when $p$ increases. We therefore consider an infinite sequence of jumps $\left(\beta_{u,i}^*\right)_{i\geq 1}$, and letting $\bar{\beta}_k^2 = 1/k \sum_{i=1}^k (\beta_{u,i}^*)^2$, we assume that $\bar{\beta}^2 = \lim_{k\to\infty} \bar{\beta}_k^2$ exists and is finite. We first show that, as $p$ increases, the first selected change-point is always the given by the same formula.

**Lemma 3.** *Assume, without loss of generality, that $u \geq n/2$. When $p \to +\infty$, the first change-point selected is*

$$\hat{u} = \underset{i\in[1,u]}{\operatorname{argmax}} \ \bar{\beta}^2 \frac{i^2 (n-u)^2}{n^2} + \sigma^2 \frac{i (n-i)}{n} \ . \tag{5}$$

*with probability tending to 1.*

From this we easily deduce under which condition the correct change point is selected, i.e., when $\hat{u} = u$:

**Theorem 4.** *Let $\alpha = u/n$ and*

$$\tilde{\sigma}_\alpha^2 = n\bar{\beta}^2 \frac{(1-\alpha)^2(\alpha - \frac{1}{2n})}{\alpha - \frac{1}{2} - \frac{1}{2n}} \ . \tag{6}$$

*When $\sigma^2 < \tilde{\sigma}_\alpha^2$, the probability that the first selected change-point is the correct one tends to $1$ as $p \to +\infty$. When $\sigma^2 > \tilde{\sigma}_\alpha^2$, it is not the correct one with probability tending to 1.*

This theorem, whose proof along with that of Lemma 3 can be found in Supplementary Materials, deserves several comments.

- To detect a change-point at position $u = \alpha n$, the noise level $\sigma^2$ must not be larger than the critical value $\sigma_\alpha$ given by (7), hence the method is not consistent for all positions. $\sigma_\alpha$ increases monotonically from $\alpha = 1/2$ to 1, meaning that change-points near the boundary are more difficult to detect correctly than change-points near the center. The most difficult change point is the last one ($u = n - 1$) which can only be detected consistently if $\sigma^2$ is smaller than

$$\bar{\sigma}_{1-1/n}^2 = \frac{2\bar{\beta}^2}{n} + o(n^{-1}).$$

- For a given level of noise $\sigma^2$, change-point detection is asymptotically correct for any $\alpha \in [\epsilon, 1 - \epsilon]$, where $\epsilon$ satisfies $\sigma^2 = \bar{\sigma}_{1-\epsilon}^2$, i.e.,

$$\epsilon = \sqrt{\frac{\sigma^2}{2n\bar{\beta}^2}} + o(n^{-1/2}) \ .$$

    This shows in particular that increasing the profile length $n$ increases the interval where change-points are correctly identified, and that we can get as close as possible to the boundary for $n$ large enough.

- When $\sigma^2 < \sigma_\alpha^2$ then the correct change-point is found consistently when $p$ increases, showing the benefit of the accumulation of many profiles.

- It is possible to make the detection of the first change-point consistent uniformly over the full signal, by simply subtracting the term $p\sigma^2 i(n-i)/n$ from $\| \bar{c}_{i,\bullet} \|^2$, which is maximized over $i$ to select the first change-point. Then, a simple modification of Lemma 3 shows that, as $p \to +\infty$, any given change-point is a.s. found. However, this modification, easy to do for the first change-point, is not obvious to extend to successive change-points detected by group LARS. We consider it an interesting future challenge to develop variants of the group LARS iterative segmentation method whose performance does not depend on the position of the change points.

## 5.2 Consistent estimation of a single change-point with fluctuating position

An interesting variant of the problem of detecting a change-point common to many profiles is that of detecting a change-point with similar location in many profiles, allowing fluctuations in the precise

location of the change-point. This can be modeled by assuming that the profiles are random, and that the $i$-th profile has a change-point of value $\beta_i$ at position $U_i$, where $(\beta_i, U_i)_{i=1,\ldots,p}$ are independent and identically distributed according to a distribution $P = P_\beta \otimes P_U$ (i.e., we assume $\beta_i$ independent from $U_i$). We denote $\bar{\beta}^2 = E_{P_\beta}\beta^2$ and $p_i = P_U(U = i)$ for $i \in [1, n-1]$. Assuming that the support of $P_U$ is $[a, b]$ with $1 \leq a \leq b \leq n-1$, the following result extends Theorem 4 by showing that, under a condition on the noise level, the first change-point discovered is indeed in the support of $P_U$:

**Theorem 5.** *Let $\alpha = U/n$ be the random position of the change-point on $[0, 1]$ and $\alpha_m = a/n$ and $\alpha_M = b/n$ the position of the left and right boundaries of the support of $P_U$ scaled to $[0, 1]$. Let also*

$$\tilde{\sigma}_{P_U}^2 = n\bar{\beta}^2 \frac{\left[(1 - E\alpha)^2 + var(\alpha)^2\right](\alpha_m - \frac{1}{2n})}{\alpha_m - \frac{1}{2} - \frac{1}{2n}} . \tag{7}$$

*If $1/2 \in (\alpha_m, \alpha_M)$, then for any $\sigma^2$ the probability that the first selected change-point is in the support of $P$ tends to 1 as $p \to +\infty$. If $1/2 < \alpha_m$, then the probability that the first selected change-point is in the support of $P$ tends to 1 when $\sigma^2 < \tilde{\sigma}_\alpha^2$, . When $\sigma^2 > \tilde{\sigma}_\alpha^2$, it is not the correct one with probability tending to 1.*

This theorem, whose proof is postponed to Supplementary Materials, illustrates the robustness of the method to handle fluctuations in the precise position of the change-point shared between the profiles. Although this situation rarely occurs when we are considering classical multidimensional signals such as financial time series or video signals, it is likely to be the rule when we consider profiles coming from different biological samples. Although the theorem only gives a condition on the noise level to ensure that the selected change-point lies in the support of the distribution of change-point locations, a precise estimate of the location of the selected change-point as a function of $P_U$, which generalizes Lemma 3, is given in the proof.

## 5.3    The case of multiple change-points

While the theoretical results presented above focus on the detection of a single change-point, the real interest of the method is to estimate multiple change-points. The extension of Theorem 4 to this setting is beyond the scope of this paper, and is postponed for future efforts. We nevertheless conjecture here that we can consistently estimate multiple change-points under conditions on the level of noise (not too large), the distance between them (not to small), and the correlations between their jumps (not too large). Indeed, following the ideas in the proof of Theorem 4, we must analyze the path of the vectors $(\hat{c}_{i,\ldots})$, and check that, for some $\lambda$ in (2), they reach their maximum norm precisely at the true change-points. The situation is more complicated than in the single change-point case since the vectors $(\hat{c}_{i,\ldots})$ must hit a hypersphere at each correct change-point, and must remain strictly within the hypersphere between consecutive change-points. This can be ensured if the noise level is not too high (like in the single change-point case), and if the positions corresponding to successive change-points on the hypersphere are far enough from each other. In practice this translates to conditions that two successive change-points should not be too close to each other, and that profiles should have, if possible, independent jumps (direction, etc.). We provide experimental results below that confirm that, when the noise is not too large, we can indeed correctly identify several change-points, with a probability of success increasing to 1 as $p$ increases.

## 6    Experiments

In this section we give experimental evidence both for theoretical $O(npk)$ complexity and Theorem 4. Figure 1 shows linearity in each of $p$, $n$ and $k$ respectively whilst fixing the other two variables, confirming the $O(npk)$ complexity.

To test Theorem 4, we considered signals of length 100, each with a unique change-point located at position $u$. We fixed $\alpha = 0.8$; assuming for simplicity that each signal jumps a height of 1 at the change-point, we get $\bar{\beta}^2 = 1$, and it is then easy to calculate the critical value $\tilde{\sigma}_\alpha^2 = 10.78$. We set the variance of the centered Gaussian noise added to each signal to $\tilde{\sigma}_\alpha^2$, and ran 1000 trials for each $u$. we expect that for $50 \leq u < 80$ there is convergence in accuracy to 1, and for $u > 80$, convergence in accuracy to zero. This is indeed what is seen in Figure 2 (left panel), with $u = 80$ the limit case between the two different modes of convergence.

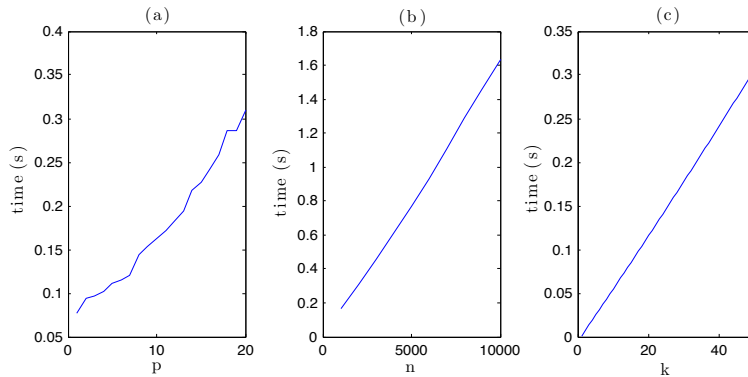

Figure 1: **Speed trials.** (a) CPU time for finding 50 change-points when there are 2000 probes and the number of profiles varies from 1 to 20. (b) CPU time when finding 50 change-points with the number of profiles fixed at 20 and the number of probes varying from 1000 to 10000 in intervals of 1000. (c) CPU time for 20 profiles and 2000 probes when selecting from 1 to 50 change-points.

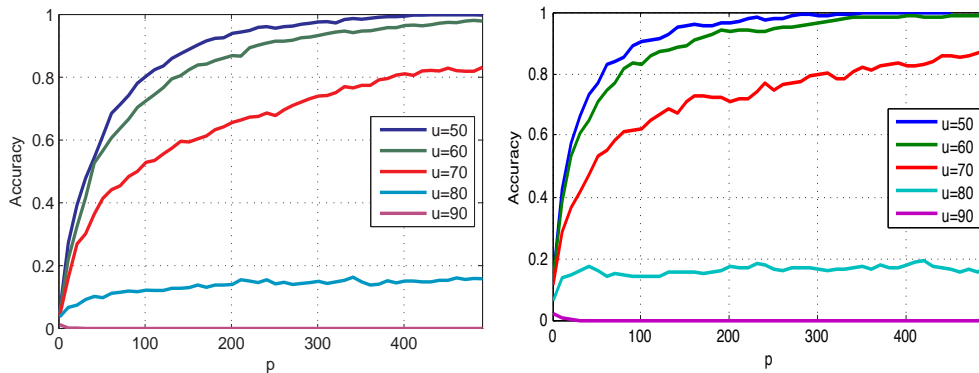

Figure 2: **Single change-point accuracy.** Accuracy as a function of the number of profiles $p$ when the change-point is placed in a variety of positions from: $u = 50$ to $u = 90$ (left panel), or: $u = 50 \pm 2$ to $u = 90 \pm 2$ (right panel), for a signal of length 100.

The right-hand-side panel of Figure 2 shows results for the same trials except that change-point locations can vary uniformly in the interval $u \pm 2$. As predicted by Theorem 5, we see that the accuracy of the method remains extremely robust against fluctuations in the exact change-point location.

To investigate the potential for extending the results of the article to the case of many shared change-points, we further simulated profiles of length 100 with a change-point at all of positions $10, 20, \ldots, 90$. The jump at each change-point was drawn from a centered Gaussian with variance 1. We then fixed various values of $\sigma^2$ and looked at convergence in accuracy as the number of signals increased. One thousand trials were performed for each $\sigma^2$, and results are presented in Figure 3. Denoting $\alpha$ the set of change-point locations $\{10, 20, \ldots, 90\}$, it appears that a critical value $\tilde{\sigma}_\alpha^2$ exists and lies close to 0.27; below 0.27 we have convergence in accuracy to 1, and above, convergence to zero.

An interesting application of the fast group LARS method is in the joint segmentation of copy-number profiles. For a set of individuals with the same disease (e.g. a type of cancer), we expect there to be regions of the genome which are frequently gained (potentially containing oncogenes) or lost (potentially containing tumor suppressor genes) in many or all of the patients. These regions are separated by change-points. Figure 4 shows Chromosome 8 of three bladder cancer copy-number profiles. We see that in the region of probe 60, a copy number change occurs on all three profiles. Though it is not in exactly the same place on all profiles, the sharing of information across profiles

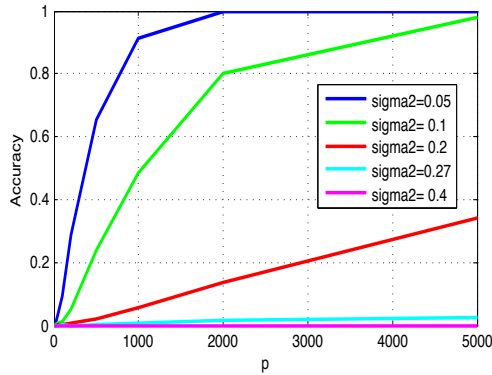

Figure 3: **Multiple change-point accuracy.** Accuracy as a function of the number of profiles $p$ when change-points are placed at the nine positions $\{10, 20, \ldots, 90\}$ and the value of $\sigma^2$ is varied from 0.1 to 0.4. The profile length is 100.

allows the approximate location to be found. The bottom right panel shows the smoothed profiles superimposed on the same axes. A promising use of these smoothed signals, beyond visualization of many profiles simultaneously, is to detect regions of frequent gain of loss by testing the average profile values on each segment for significant positive (gain) or negative (loss) values. Preliminary experiments on simulated and real data suggest that our method is more accurate and two orders of magnitude faster than the state-of-the-art H-HMM [19] method for that purpose.

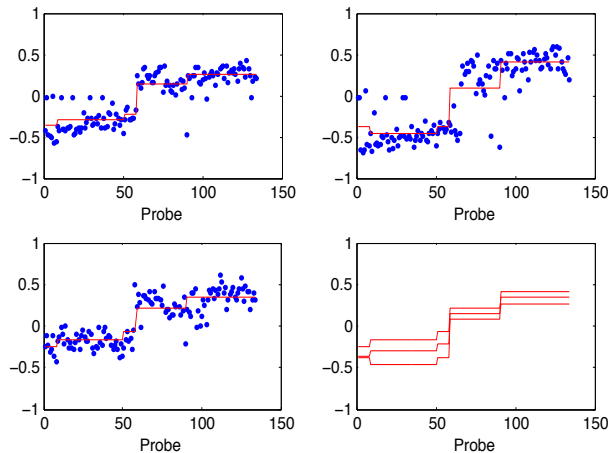

Figure 4: **Segmented and smoothed bladder cancer copy-number profiles.** Probes shown are located on Chromosome 8. A shared change-point hotspot is found in the region of probe 60.

# 7 Conclusion

We have proposed a framework that extends total-variation based approximation to the multidimensional setting, developed a fast algorithm to approximately solve it, shown theoretically that the method can consistently estimate change-points, and validated the results experimentally. We have not discussed the problem of choosing the number of change-points, and suggest in practice to use existing criteria for this purpose [6, 7]. We observed both theoretically and empirically that increasing the number of profiles is highly beneficial to detect shared change-points

**Acknowledgements** We thank Zaid Harchaoui and Francis Bach for useful discussions. This work was supported by ANR grants ANR-07-BLAN-0311-03 and ANR-09-BLAN-0051-04.

# References

[1] Z. Harchaoui, F. Vallet, A. Lung-Yut-Fong, and O. Cappe. A regularized kernel-based approach to unsupervised audio segmentation. In *ICASSP '09: Proceedings of the 2009 IEEE International Conference on Acoustics, Speech and Signal Processing*, pages 1665–1668, Washington, DC, USA, 2009. IEEE Computer Society.

[2] N. R. Zhang, D. O. Siegmund, H. Ji, and J. Li. Detecting simultaneous change-points in multiple sequences. *Biometrika*, 97(3):631–645, 2010.

[3] M. Basseville and N. Nikiforov. *Detection of abrupt changes: theory and application*. Information and System Sciences Series. Prentice Hall Information, 1993.

[4] B. Brodsky and B. Darkhovsky. *Nonparametric Methods in Change-Point Problems*. Kluwer Academic Publishers, 1993.

[5] Y. C. Yao. Estimating the number of change-points via schwarz criterion. *Stat. Probab. Lett.*, 6:181–189, 1988.

[6] L. Birgé and P. Massart. Gaussian model selection. *J. Eur. Math. Soc.*, 3:203–268, 2001.

[7] M. Lavielle and G. Teyssière. Detection of multiple change-points in multivariate time series. *Lithuanian Mathematical Journal*, 46(3):287–306, 2006.

[8] L. J. Vostrikova. Detection of disorder in multidimensional stochastic processes. *Soviet Mathematics Doklady*, 24:55–59, 1981.

[9] M. Lavielle and Teyssière. Adaptive detection of multiple change-points in asset price volatility. In G. Teyssière and A. Kirman, editors, *Long-Memory in Economics*, pages 129–156. Springer Verlag, Berlin, 2005.

[10] L. I. Rudin, S. Osher, and E. Fatemi. Nonlinear total variation based noise removal algorithms. *Physica D*, 60:259–268, 1992.

[11] R. Tibshirani, M. Saunders, S. Rosset, J. Zhu, and K. Knight. Sparsity and smoothness via the fused lasso. *J. R. Stat. Soc. Ser. B Stat. Methodol.*, 67(1):91–108, 2005.

[12] Z. Harchaoui and C. Levy-Leduc. Catching change-points with lasso. In J.C. Platt, D. Koller, Y. Singer, and S. Roweis, editors, *Advances in Neural Information Processing Systems 20*, pages 617–624. MIT Press, Cambridge, MA, 2008.

[13] A. Rinaldo. Properties and refinements of the fused lasso. *Ann. Stat.*, 37(5B):2922–2952, 2009.

[14] J. Friedman, T. Hastie, H. Höfling, and R. Tibshirani. Pathwise coordinate optimization. *Ann. Appl. Statist.*, 1(1):302–332, 2007.

[15] H. Hoefling. A path algorithm for the Fused Lasso Signal Approximator. Technical Report 0910.0526v1, arXiv, Oct. 2009.

[16] M. Yuan and Y. Lin. Model selection and estimation in regression with grouped variables. *J. R. Stat. Soc. Ser. B*, 68(1):49–67, 2006.

[17] J. Mairal, F. Bach, J. Ponce, and G. Sapiro. Online learning for matrix factorization and sparse coding. *J. Mach. Learn. Res.*, 11:19–60, 2010.

[18] S. Boyd and L. Vandenberghe. *Convex Optimization*. Cambridge University Press, New York, NY, USA, 2004.

[19] S.P. Shah, W.L. Lam, R.T. Ng, and K.P. Murphy. Modeling recurrent DNA copy number alterations in array CGH data. *Bioinformatics*, 23(13):i450–i458, 2007.

